# A rational model of preference learning and choice prediction by children

**Christopher G. Lucas**
Department of Psychology
University of California, Berkeley
Berkeley, CA 94720, USA
clucas@berkeley.edu

**Thomas L. Griffiths**
Department of Psychology
University of California, Berkeley
Berkeley, CA 94720, USA
tom_griffiths@berkeley.edu

**Fei Xu**
Department of Psychology
University of British Columbia
Vancouver, B.C., Canada V6T 1Z4
fei@psych.ubc.ca

**Christine Fawcett**
Max-Planck-Institute for Psycholinguistics
Wundtlaan 1, Postbus 310
6500AH, Nijmegen, The Netherlands
christine.fawcett@mpi.nl

## Abstract

Young children demonstrate the ability to make inferences about the preferences of other agents based on their choices. However, there exists no overarching account of what children are doing when they learn about preferences or how they use that knowledge. We use a rational model of preference learning, drawing on ideas from economics and computer science, to explain the behavior of children in several recent experiments. Specifically, we show how a simple econometric model can be extended to capture two- to four-year-olds' use of statistical information in inferring preferences, and their generalization of these preferences.

## 1 Introduction

Economists and computer scientists are often concerned with inferring people's preferences from their choices, developing econometric methods (e.g., [1, 2]) and collaborative filtering algorithms (e.g., [3, 4, 5]) that will allow them to assess the subjective value of an item or determine which other items a person might like. However, identifying the preferences of others is also a key part of social cognitive development, allowing children to understand how people act and what they want. Young children are thus often in the position of economists or computer scientists, trying to infer the nebulous preferences of the people around them from the choices they make. In this paper, we explore whether the inferences that children draw about preferences can be explained within the same kind of formalism as that used in economics and computer science, testing the hypothesis that children are making rational inferences from the limited data available to them.

Before about 18 months of age, children seem to assume that everyone likes the same things as themselves, having difficulty understanding the subjective nature of preferences (e.g., [6]). However, shortly after coming to recognize that different agents can maintain different preferences, children demonstrate a remarkably sophisticated ability to draw conclusions about the preferences of others from their behavior. For example, two-year-olds seem to be capable of using shared preferences between an agent and themselves as the basis for generalization of other preferences [7], while three- and four-year-olds can use statistical information to reason about preferences, inferring a preference for an object when an agent chooses the object more often than expected by chance [8].

This literature in developmental psychology is paralleled by work in econometrics on statistical models for inferring preferences from choices. In this paper, we focus on an approach that grew out

of the Nobel prize-winning work of McFadden (see [1] for a review), exploring a class of models known as mixed multinomial logit models [2]. These models assume that agents assign some utility to every option in a choice, and choose in a way that is stochastically related to these utilities. By observing the choices people make, we can recover their utilities by applying statistical inference, providing a simple rational standard against which the inferences of children can be compared.

Research on preferences in computer science has tended to go beyond modeling individual choice, focusing on predicting which options people will like based not just on their own previous choice patterns but also drawing on the choices of other people – a problem known as *collaborative filtering* [3]. This work has led to the development of the now-ubiquitous recommendation systems that suggest which items one might like to purchase based on previous purchases, and has reached notoriety through the recent Netflix challenge. Economists have also explored models for the choices of multiple agents, using hierarchical Bayesian statistics [9]. These models combine information across agents to make inferences about the properties or value of different options.

Our contribution in this paper is to bring together these different threads of research to develop rational models of children's inferences about preferences. Section 2 summarizes developmental work on children's inferences about preferences. Section 3 outlines the basic idea behind rational choice models, drawing on previous work in economics and computer science. We then consider how these models can be used to explain developmental data, with Section 4 concerned with inferences about preferences from choices, and Section 5 focusing on inferences about the properties of objects from preferences. Section 6 discusses the implications of our results and concludes the paper.

## 2  Children's inferences about preferences

The basic evidence that children do not differentiate the preferences of others before about 18 months of age comes from Repacholi and Gopnik [6]. Subsequent work has built on these results to explore the kinds of cues that children can use in inferring preferences, and how children generalize consistent patterns of preferences.

### 2.1  Learning preferences from statistical evidence

While 18-month-olds are able to infer preferences from affective responses, we often need to make inferences from more impoverished data, such as the patterns of choices that people make when faced with various options. Recent work by Kushnir and colleagues [8] provided the first evidence that 3- and 4-year-old children can use statistical sampling information as the basis for inferring an agent's preference for toys. Three groups of children were tested in a simple task. Each child was shown a big box of toys. For the first group, the box was filled with just one type of toy (e.g., 100% red discs). For the second group, the box was filled with two types of toys (e.g., 50% red discs and 50% blue plastic flowers). For the third group, the box was also filled with two types of toys, but in different proportions (e.g., 18% red discs and 82% blue plastic flowers). A puppet named Squirrel came in to play a game with the child. Squirrel looked into the box and picked out five toys. The sample always consisted of five red discs for all three conditions. Then the child was given three toys – a red disc (the target), a blue plastic flower (the alternative), and a yellow cylinder (the distractor) – and was asked to give Squirrel the one he liked. Each child received two trials with different objects. The results of the experiment showed that the children chose the target (the red disc) 0.96, 1.29, and 1.67 times (out of 2) in the 100%, 50%, and 18% conditions, respectively, suggesting that children used the non-random sampling behavior of Squirrel as the basis for inferring his preferences.

### 2.2  Generalizing from shared preferences

Recognizing that preferences can vary from one agent to another also establishes an opportunity to discover that those preferences can differ in the degree to which they are related to one's own. Fawcett and Markson [7] asked under what conditions children would use shared preferences between themselves and another agent as the basis for generalization, using a task similar to the "collaborative filtering" problem explored in computer science. Their experiments began with four blocks of training involving two actors. In each block the actors introduced two objects from a common category, including toys, television shows and foods. Each actor expressed liking the object she introduced and dislike for the other's object. One actor had preferences that were matched to the

child's in all blocks, in that her objects had features chosen to be more interesting to the child. After each actor reacted to the objects, the child was given an opportunity to play with the objects, and his or her preference for one object over the other was judged by independent coders, based on relative interest in and play with each object.

After the training blocks, the first test block began. Each actor brought out a new object that was described as being in the same category as the training objects, but was hidden from the child by an opaque container. Each actor then reacted to her novel object in a manner that varied by condition. In the *like* condition, the actor's reaction was to examine the object and describe it as her favorite object of the category. In the *dislike* condition, she examined the object and expressed dislike. In the *indifferent* condition the actor did not examine the toy, and professed ignorance about it. The child was then given an opportunity to choose one hidden object to play with. Finally, a second test block began, identical to the first except that the hidden objects were members of a different category from those seen in training. In Experiment 1, members of the new category could be taken to share features with members of the training category, e.g., toys versus books, while in Experiment 2 the new category was chosen to minimize such overlap, e.g., food versus television shows. Children consistently chose the test items that were favored by the agent who shared their own preferences during training, for both toys and the similar category, books. In contrast, when a highly distant category was used during test, children did not show any systematic generalization behaviors. These results suggest that children use shared preferences as the basis for generalization, but they also take into account whether the categories are related or not.

### 2.3 Summary and prospectus

Recent results in developmental psychology indicate that young children are capable of making remarkably sophisticated inferences about the preferences of others. This raises the question of how they make these inferences, and whether the kinds of conclusions that children draw from the behavior of others are justified. We explore this question in the remainder of the paper. In the tradition of rational analysis [10], we consider the problem of how one might *optimally* infer people's preferences from their choices, and compare the predictions of such a model with the developmental data. The results of this analysis will help us understand how children might conceive of the relationship between the choices that people make and the preferences they have.

## 3 A rational model connecting choice and preference

In developing a rational account of how an agent might learn others' preferences from choice information, we must first posit a specific ecological relationship between people's preferences and their choices, and then determine how an agent would make optimal inferences from others' behavior given knowledge of this relationship. Fortunately, the relationship between preferences and choices has been the subject of extensive research in economics and psychology.

One of the most basic models of choice behavior is the Luce-Shepard choice rule [11, 12], which asserts that when presented with a set of $J$ options with utilities $\mathbf{u} = (u_1, \ldots, u_J)$, people will choose option $i$ with probability

$$P(c = i | \mathbf{u}) = \frac{\exp(u_i)}{\sum_j \exp(u_j)} \qquad (1)$$

where $j$ ranges over the options considered in the choice. Given this choice rule, learning about an agent's preferences is a simple matter of Bayesian inference. Specifically, having observed a sequence of choices $\mathbf{c} = (c_1, \ldots, c_N)$, we can compute a posterior distribution over the utilities of the options involved by applying Bayes' rule

$$p(\mathbf{u} | \mathbf{c}) = \frac{P(\mathbf{c} | \mathbf{u}) p(\mathbf{u})}{\int P(\mathbf{c} | \mathbf{u}) p(\mathbf{u}) \, d\mathbf{u}} \qquad (2)$$

where $p(\mathbf{u})$ is a probability density expressing the prior probability of a vector of utilities $\mathbf{u}$, and the likelihood $P(\mathbf{c} | \mathbf{u})$ is obtained by assuming that the choices are independent given $\mathbf{u}$, being the product of the probabilities of the individual choices as in Equation 1.

While this simple model is sufficient to capture preferences among a constrained set of objects, most models used in econometrics aim to predict the choices that agents will make about novel objects.

This can be done by assuming that options have features that determine their utility, with the utility of option $i$ being a function of the utility of its features. If we let $\mathbf{x}_i$ be a binary vector indicating whether an option possesses each of a finite set of features, and $\boldsymbol{\beta}_a$ be the utility that agent $a$ assigns to those features[1], we can express the utility of option $i$ for agent $a$ as the inner product of these two vectors. The probability of agent $a$ choosing option $i$ is then

$$P(c = j|\mathbf{X}, \boldsymbol{\beta}_a) = \frac{\exp(\boldsymbol{\beta}_a^T \mathbf{x}_i)}{\sum_j \exp(\boldsymbol{\beta}_a^T \mathbf{x}_j)} \tag{3}$$

where $\mathbf{X}$ collects the features of all of the options. We can also integrate out $\boldsymbol{\beta}_a$ to obtain the choice probabilities given just the features of the options, with

$$P(c = j|\mathbf{X}) = \int \frac{\exp(\boldsymbol{\beta}_a^T \mathbf{x}_i)}{\sum_j \exp(\boldsymbol{\beta}_a^T \mathbf{x}_j)} p(\boldsymbol{\beta_a}) \, d\boldsymbol{\beta}_a \tag{4}$$

This corresponds to the mixed multinomial logit (MML; [2]) model, which has been used for several decades in econometrics to model discrete-choice preferences in populations of consumers.

The MML model and the Luce-Shepard rule on which it is built are theoretically appealing for several reasons. First, the Luce-Shepard rule reflects the choice probabilities that result when agents seek to maximize their utility in the presence of noise on utilities that follows a Weibull distribution [13], and is thus compatible with the standard assumptions of statistical decision theory. Second, the MML can approximate the distribution of choices for essentially any heterogeneous population of utility-maximizing agents given appropriate choice of $p(\boldsymbol{\beta}_a)$ [1]. Finally, this approach has been widely used and generally successful in applications in marketing and econometrics (e.g., [9, 14]).

While a wide range of random utility models can be represented with an appropriate choice of prior over $\boldsymbol{\beta}_a$, one common variation supposes that $\boldsymbol{\beta}_a$ follows a Gaussian distribution around a population mean which in turn has a Gaussian prior. Given that the individuals in the experiments we examine are only dealing with two actors, we will assume a single-parameter prior in which different agents' preferences are independent and preferences for individual features are uncorrelated with a Gaussian distribution with mean zero and variance $\sigma^2$: $\boldsymbol{\beta}_a \sim \mathcal{N}(0, \sigma^2 \mathbf{I})$.

The model outlined in this section provides a way to optimally answer the question of how to infer the preferences of an agent from their choices. In the remainder of the paper, we explore how well this simple rational model accounts for the inferences that children make about preferences, applying the model to the key developmental phenomena introduced in the previous section.

## 4   Using statistical information to infer preferences

The experiment conducted by Kushnir and colleagues [8], discussed in Section 2.1, provides evidence that children are sensitive to statistical information when inferring the preferences of agents. In this section, we examine whether this inference is consistent with the predictions of the rational model outlined above. We first consider how to apply the MML model in this context, then discuss the model predictions and alternative explanations.

### 4.1   Applying the MML model

The child's goal is to learn what Squirrel's preferences are, so as to offer an appropriate toy. Let $\boldsymbol{\beta}_a$ be Squirrel's preferences, $\mathbf{c} = (c_1 \ldots c_N)$ the sequence of $N$ choices Squirrel makes, and $\mathbf{X}_n = [\mathbf{x}_{n1} \ldots \mathbf{x}_{nJ_n}]^T$ the observed features of Squirrel's $J_n$ options at choice event $n$. The set $\{\mathbf{X}_1, \ldots, \mathbf{X}_N\}$ will be denoted with $\mathbf{X}$. Estimating $\boldsymbol{\beta}_a$ entails computing $p(\boldsymbol{\beta}_a|\mathbf{c}, \mathbf{X}) \propto P(\mathbf{c}|\boldsymbol{\beta}_a, \mathbf{X})p(\boldsymbol{\beta}_a)$, analogous to the inference of $\mathbf{u}$ in Equation 2. The probability of Squirrel's choices is $P(\mathbf{c}|\mathbf{X}, \boldsymbol{\beta}_a) = \prod_{n=1}^N P(c_n|\mathbf{X}_n, \boldsymbol{\beta}_a)$, where $P(c_n = j|\mathbf{X}_n, \boldsymbol{\beta}_a)$ is given by Equation 3.

We chose to represent the objects as having minimal and orthogonal feature vectors, so that red discs (Squirrel's target toy) had features $[1\ 0\ 0]^T$, blue flowers (the alternative option in his choices) had features $[0\ 1\ 0]^T$, and yellow cylinders (the distractor) had features $[0\ 0\ 1]^T$, respectively. The

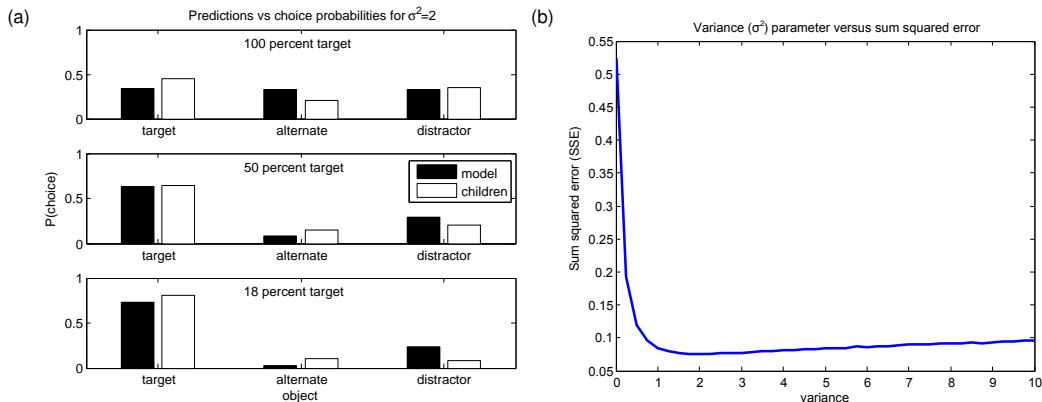

Figure 1: Model predictions for data in [8]. (a) Predicted probability that objects will be selected, plotted against observed proportions. (b) Sensitivity of model to setting of variance parameter.

number of options in each choice Squirrel made from the box was the total number of objects in the box (38), with each type of object being represented with the appropriate frequency. The $N = 5$ choices made by Squirrel thus provide the data $\mathbf{c}$ from which its preferences can be inferred. We constructed an approximation to the posterior distribution over $\boldsymbol{\beta}_a$ given $\mathbf{c}$ using importance sampling (see, e.g., [15] for details), drawing a sample of values of $\boldsymbol{\beta}_a$ from the prior distribution $p(\boldsymbol{\beta}_a)$ and giving each value weight proportional to the corresponding likelihood $P(\mathbf{c}|\mathbf{X}, \boldsymbol{\beta}_a)$.

The child must now select an object to give Squirrel, with one target, one alternative, and one distractor as options. If we suppose each child is matching Squirrel's choice distribution, we can use the Luce-Shepard choice rule (Equation 3) to predict the rates at which children should choose the different objects for a particular value of $\boldsymbol{\beta}_a$, $P(c_{\text{child}} = j|\mathbf{X}, \boldsymbol{\beta}_a)$. The probability of a particular choice is then obtained by averaging over $\boldsymbol{\beta}_a$, with

$$P(c_{\text{child}} = j|\mathbf{X}, \mathbf{c}) = \int P(c_{\text{child}} = j|\mathbf{X}, \boldsymbol{\beta}_a)p(\boldsymbol{\beta}_a|\mathbf{X}, \mathbf{c}) \, d\boldsymbol{\beta}_a \qquad (5)$$

which we compute using the approximate posterior distribution yielded by importance sampling. All simulations presented here use $10^6$ samples, and were performed for a range of values of $\sigma^2$, the parameter that determines the variance of the prior on $\boldsymbol{\beta}_a$.

## 4.2   Results

Figure 1 (a) compares the predictions of the model with $\sigma^2 = 2$ to the participants' choice probabilities. The sum squared error (SSE) of the predictions was .0758 when compared with the observed probabilities of selecting the target object, with the correlation of the model predictions and observed data being $r = 0.93$. Figure 1 (b) shows that the goodness of fit is generally insensitive to the variance of the prior $\sigma^2$, provided $\sigma^2 > 1$. This is essentially the only free parameter of the model, as it sets the scale for the features $\mathbf{X}$, indicating that there is a close correspondence between the predictions of the rational model and the inferences of the children under a variety of reasonable assumptions about the distribution of preferences.

The only conspicuous difference between the model's predictions and the children's choices was the tendency of children to choose the target object more frequently than alternatives even when it was the only object in the box. This can be explained by observing that under the cover story used in the experiment, Squirrel was freely choosing to select objects from the box, implicitly indicating that it was choosing these objects over other unobserved options. As a simple test of this explanation, we generated new predictions under the assumption that each choice included one other unobserved option, with features orthogonal to the choices in the box. This improved the fit of the model, resulting in an SSE of .05 and a correlation of $r = .95$.

### 4.3 Alternative explanations

Kushnir and colleagues [8] suggest that the children in the experiment may be learning preferences by using statistical information to identify situations where the agent's behavior is not consistent with random sampling. We do not dispute that this may be correct; our analysis does not entail a commitment to a procedure by which children make inferences about preferences, but to the idea that whatever the process is, it should provides a good solution to the problem with which children are faced given the constraints under which they operate. We will add two observations. The first is that we need to consider how such an explanation might be generalized to explain behavior in other preference-learning situations, and if not, what additional processes might be at work. The second is that it is not difficult to test salient variations on the sampling-versus-preference view that are inconsistent with our own, as they predict that children make a dichotomous judgment – distinguishing random from biased sampling – rather than one that reveals the extent of a preference in addition to its presence and valence. The former predicts that over a wide range of sets of evidence that indicate an agent strongly prefers objects of type X to those of type Y, one can generate evidence consistent with a *weaker* preference for type W over Y (over more data points) that will lead children to offer the agent objects of type W over X.

## 5 Generalizing preferences to novel objects

The study of Fawcett and Markson [7] introduced in Section 2.2 provides a way to go beyond simple estimation of preferences from choices, exploring how children solve the "collaborative filtering" problem of generalizing preferences to novel objects. We will outline how this can be captured using the MML model, present simulation results, and then consider alternative explanations.

### 5.1 Applying the MML model

Forming an appropriate generalization in this task requires two kinds of inferences. The first inference the child must make – learning the actors' preferences by computing $p(\boldsymbol{\beta}_a|\mathbf{X}, \mathbf{c})$ for $a \in (1, 2)$ – is the same as that necessary for the first set of experiments discussed above. The second inference is estimating the two hidden objects' features via those preferences. In order to solve this problem, we need to modify the model slightly to allow us to predict actions other than choices. Specifically, we need to define how preferences are related to affective responses, since the actors simply indicated their affective response to the novel object.

We will refer to the actor whose preferences matched those of the child as Actor 1, and the other actor as Actor 2. Let the features of Actor $a$'s preferred object in round $n$ be $\mathbf{x}_{na}$, the features of the same-category novel object be $\mathbf{x}_{sa}$, and the features of the different-category novel object be $\mathbf{x}_{da}$. When the category is irrelevant, we will use $\mathbf{x}_{*a} \in \{\mathbf{x}_{sa}, \mathbf{x}_{da}\}$ to indicate the features of the novel object. The goal of the child is to infer $\mathbf{x}_{*a}$ (and thus whether they themselves will like the novel object) from the observed affective response of agent $a$, the features of the objects from the previous rounds $\mathbf{X}$, and the choices of the agent on the previous rounds $\mathbf{c}$. This can be done by evaluating

$$P(\mathbf{x}_{*a}|\mathbf{X}, \mathbf{c}, r_a) = \int P(\mathbf{x}_{*a}|\boldsymbol{\beta}_a, r_a)p(\boldsymbol{\beta}_a|\mathbf{X}, \mathbf{c}) \, d\beta_a \tag{6}$$

where $P(\mathbf{x}_{*a}|\boldsymbol{\beta}_a, r_a)$ is the posterior distribution over the features of the novel object given the preferences and affective response of the agent. Computing this distribution requires defining a likelihood $P(r_a|\mathbf{x}_{*a}, \boldsymbol{\beta}_a)$ and a prior on features $P(\mathbf{x}_{*a})$. We deal with these problems in turn.

The likelihood $P(r_a|\mathbf{x}_{*a}, \boldsymbol{\beta}_a)$ reflects the probability of the agent producing a particular affective response given the properties of the object and the agent's preferences. In the experiment, the affective responses produced by the actors were of two types. In the *like* condition, the actor declared this to be her favorite object. If one takes the actor's statement at face value and supposes that the actor has encountered an arbitrarily large number of such objects, then

$$P(\mathbf{x}_{*a}|\boldsymbol{\beta}_a, r_a = \text{"like"}) = 1 \text{ for } \mathbf{x}_{*a} = \arg \max_{\mathbf{x}_{*a}} \boldsymbol{\beta}_a^T \mathbf{x}_{*a}, \text{else } 0 \tag{7}$$

In the *dislike* condition, the action – saying "there's a toy in here, but I don't like it"– communicates negative utility, or at least utility below some threshold which we will take to be zero, hence:

$$P(\mathbf{x}_{*a}|\boldsymbol{\beta}_a, r_a = \text{"dislike"}) = 1 \text{ for } \boldsymbol{\beta}_a^T \mathbf{x}_{*a} < 0, \text{else } 0 \tag{8}$$

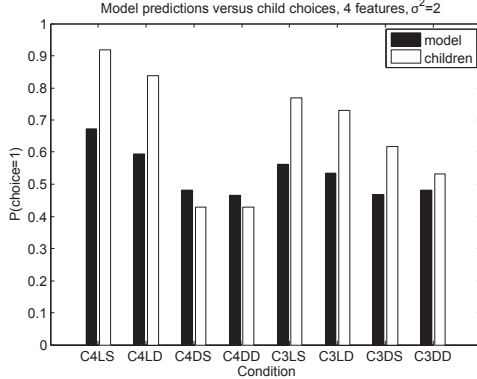

Figure 2: Model predictions for data in Experiment 1 of [7], excluding cases where children had fewer than 4 chances to play with training objects. The prefixes C4 and C3 denote cases where children chose to play with the objects presented by the matched actor (i.e., Actor 1) in training 4 and 3 times out of 4, respectively. The third character denotes the *like* (L) or *dislike* (D) condition, and the fourth character denotes whether the object is in the same category (S) or a different category (D) from those seen in training. $P(\text{choice} = 1)$ is the probability of selecting Actor 1's novel object.

In defining the prior distributions from which the features of both the observed objects and the novel objects are sampled, it is important to represent differences between categories. Our feature vectors were concatenations of category-1 features, category-2 features, and multiple-category features, where each feature was present with probability $0.5$ if its category could possess the feature, otherwise zero. We arbitrarily chose four features per category, for a total of twelve.

Having computed a posterior distribution over $\mathbf{x}_{*a}$ using this prior and likelihood, the child must combine this information with his or her own preferences to select an object. Unfortunately, we do not have direct access to the utilities of the children in this experiment, so we must estimate them from the children's choice data. We did this using the same procedure as for the adults' utilities. That done, we apply the choice rule a final time and obtain choice probabilities for the objects selected by the children on the final trial.

## 5.2 Results

Figure 2 shows the rates at which children chose actor 1's object and the model's predictions for the same. With the variance parameter set to $\sigma^2 = 2$ and twelve features, the correlation between the predictions and the data was $r = 0.88$. When examining only the 4-year-old participants ($N = 68$), the correlation rose to $r = 0.94$. The number of features has little influence on predictive accuracy: with 30 features, the correlations were $r = 0.85$ and $r = 0.93$ for all participants and 4-year-old participants, respectively. The model predicts less-extreme probabilities than were observed in the choices of the children, in particular in the cases where children chose to play with one of actor 2's objects. This may be attributed to actor 1's objects having features one would expect people to like a priori, making the zero-mean preference prior inappropriate.

## 5.3 Alternative explanations

The model described above provides good predictions of children's inferences in this experiment, but also attributes relatively complex beliefs to the child. A natural question is whether a simpler mechanism might explain behavior in this case. Fawcett and Markson [7] discussed several alternative explanations of their findings, so this section will briefly recapitulate their discussion with a specific view towards alternative models.

The simplest model might suggest that children were simply learning associations between specific behaviors by the matching actor and the presence of a desirable object. The most basic model of this kind, assuming that the matching actor is generally associated with desirable objects, is falsified by the children showing no bias towards the matching actor's objects in the dislike condition. A more elaborate version, in which children associate positive affect by the matching actor with desirable objects, is inconsistent with children's stronger preferences when the novel objects were similar to the training objects. This evidence also runs against explanations that the children are selecting objects based on liking the matched actor or believing that the matched actor is a more reliable judge of quality. Any of these alternatives might be made to fit the data via ad hoc assumptions

about subjective feature correlations or category similarity, but we see no reason to adopt a more complex model without better explanatory power.

## 6 Conclusion

We have outlined a simple rational model for inferences about preferences from choices, drawing on ideas from economics and computer science, and shown that this model produces predictions that closely parallel the behavior of children reasoning about the preferences of others. These results shed light on how children may think about choice, desire, and other minds, and highlight new questions and possible extensions. In future work, we intend to explore whether the developmental shift discovered by Repacholi and Gopnik [6] can be explained in terms of rational model selection under an MML view. We believe that a hierarchical MML and the "Bayesian Ockham's razor" provide a simple account, resembling a recent Bayesian treatment of false-belief learning [16]. Our approach also provides a framework for predicting how children might make inferences to preferences at the population level and exploring the information provided by correlated features.

**Acknowledgments.** This work was supported by AFOSR grant number FA9550-07-1-0351, NSERC and SSHRC Canada, and the McDonnell Causal Learning Collaborative.

## Footnotes

[1]We will refer to the utilities of features as "preferences" to distinguish them from the utilities of options.

## References

[1] D. McFadden and K. E. Train. Mixed MNL models of discrete response. *Journal of Applied Econometrics*, 15:447–470, 2000.

[2] J. Hayden Boyd and R. E. Mellman. Effect of fuel economy standards on the u.s. automotive market: An hedonic demand analysis. *Transportation Research A*, 14:367–378, 1980.

[3] D. Goldberg, David N., B. M. Oki, and T. Douglas. Using collaborative filtering to weave an information tapestry. *Communications of the ACM*, 35(12):61–70, 1992.

[4] J. A. Konstan, B. N. Miller, D. Maltz, J. L. Herlocker, L. R. Gordon, and J. Riedl. Grouplens: applying collaborative filtering to usenet news. *Communications of the ACM*, 40:77–87, 1997.

[5] C. Kadie J. Breese, D. Heckerman. Empirical analysis of predictive algorithms for collaborative filtering. In *Proceedings of the Fourteenth Annual Conference on Uncertainty in Artificial Intelligence (UAI 98)*, San Francisco, CA, 1998. Morgan Kaufmann.

[6] B. M. Repacholi and A. Gopnik. Early reasoning about desires: Evidence from 14- and 18-month-olds. *Developmental Psychology*, 33(1):12–21, 1997.

[7] C. A. Fawcett and L. Markson. Children reason about shared preferences. revised manuscript submitted for publication. *Developmental Psychology*, under review.

[8] T. Kushnir, F. Xu, and H. Wellman. Preschoolers use sampling information to infer the preferences of others. In *28th Annual Conference of the Cognitive Science Society*, 2008.

[9] K. E. Train, D. McFadden, and M. Ben-Akiva. The demand for local telephone service: A fully discrete model of residential calling patterns and service choices. *The RAND Journal of Economics*, 18(1):109–123, 1987.

[10] J. R. Anderson. *The adaptive character of thought*. Erlbaum, Hillsdale, NJ, 1990.

[11] R. D. Luce. *Individual choice behavior*. John Wiley, New York, 1959.

[12] R. N. Shepard. Stimulus and response generalization: A stochastic model relating generalization to distance in psychological space. *Psychometrika*, 22:325–345, 1957.

[13] D. McFadden. Conditional logit analysis of qualitative choice behavior. In P. Zarembka, editor, *Frontiers in Econometrics*. Academic Press, New York, 1973.

[14] D. Revelt and K. E. Train. Mixed logit with repeated choices: Households' choices of appliance efficiency level. *The Review of Economics and Statistics*, 80(4):647–657, 1998.

[15] R. M. Neal. Probabilistic inference using Markov chain Monte Carlo methods. Technical Report CRG-TR-93-1, University of Toronto, 1993.

[16] N. D. Goodman, C. L. Baker, E. Baraff Bonawitz, V. K. Mansinghka, A. Gopnik, H. Wellman, L. Schulz, and J. B. Tenenbaum. Intuitive theories of mind: A rational approach to false belief. In *28th Annual Conference of the Cognitive Science Society*, 2006.

